# Forging The Graphs: A Low Rank and Positive Semidefinite Graph Learning Approach

**Dijun Luo, Chris Ding, Heng Huang, Feiping Nie**
Department of Computer Science and Engineering
The University of Texas at Arlington
dijun.luo@gmail.com, chqding@uta.edu
heng@uta.edu, feipingnie@gmail.com

## Abstract

In many graph-based machine learning and data mining approaches, the quality of the graph is critical. However, in real-world applications, especially in semi-supervised learning and unsupervised learning, the evaluation of the quality of a graph is often expensive and sometimes even impossible, due the cost or the un-availability of ground truth. In this paper, we proposed a robust approach with *convex optimization* to "forge" a graph: with an input of a graph, to learn a graph with higher quality. Our major concern is that an ideal graph shall satisfy all the following constraints: non-negative, symmetric, low rank, and positive semidefinite. We develop a graph learning algorithm by solving a convex optimization problem and further develop an efficient optimization to obtain global optimal solutions with theoretical guarantees. With only one non-sensitive parameter, our method is shown by experimental results to be robust and achieve higher accuracy in semi-supervised learning and clustering under various settings. As a prepro-cessing of graphs, our method has a wide range of potential applications machine learning and data mining.

## 1 Introduction

Many machine learning algorithms use graphs as input, such as clustering [16, 14], manifold based dimensional reduction [2, 15], and graph-based semi-supervised learning [23, 22]. In these ap-proaches, we are particularly interested in the *similarity* among objects. However, the observation of similarity graphs often contain noise which sometimes mislead the learning algorithm, especially in unsupervised and semi-supervised learning. Deriving graphs with high quality becomes attractive in machine learning and data mining research.

A robust and stable graph learning algorithm is especially desirable in unsupervised and semi-supervised learning, because the unavailability or high cost of ground truth in real world appli-cations. In this paper, we develop a novel graph learning algorithm based on convex optimization, which leads to robust and competitive results.

### 1.1 Motivation and Main Problem

In this section, the properties of similarity matrix are revisited from point of view of normalized cut clustering [19]. Given a symmetric similarity matrix $\mathbf{W} \in \mathbb{R}^{n \times n}$ on $n$ objects, normalized cut solves the following optimization problem [10].

$$\min_{\mathbf{H} \geq 0} \mathbf{tr} \mathbf{H}^\mathsf{T} (\mathbf{D} - \mathbf{W}) \mathbf{H} \ \ s.t. \ \ \mathbf{H}^\mathsf{T} \mathbf{D} \mathbf{H} = \mathbf{I}, \tag{1}$$

where $\mathbf{H} \in \{0, 1\}^{n \times K}$ is the cluster indicator matrix, or equivalently,

$$\max_{\mathbf{F} \geq 0} \mathbf{tr} \mathbf{F}^\mathsf{T} \tilde{\mathbf{W}} \mathbf{F} \quad s.t. \quad \mathbf{F}^\mathsf{T} \mathbf{F} = \mathbf{I}, \tag{2}$$

where $\mathbf{F} = [f_1, f_2, \cdots, f_K], \mathbf{H} = [h_1, h_2, \cdots, h_K], f_k = \mathbf{D}^{\frac{1}{2}} h_k / \|\mathbf{D}^{\frac{1}{2}} h_k\|, 1 \leq k \leq K, \tilde{\mathbf{W}} = \mathbf{D}^{-\frac{1}{2}} \mathbf{W} \mathbf{D}^{-\frac{1}{2}}, \mathbf{D} = \mathbf{diag}(d_1, d_2, \cdots, d_n), d_i = \sum_{j=1}^{n} W_{ij}, \mathbf{I}$ is the identity matrix, and $K$ is the number of groups. Eq. (2) can be further rewritten as,

$$\min_{\mathbf{F} \geq 0} \|\tilde{\mathbf{W}} - \mathbf{F} \mathbf{F}^\mathsf{T}\|_F \quad s.t. \quad \mathbf{F}^\mathsf{T} \mathbf{F} = \mathbf{I}, \tag{3}$$

where $\| \cdot \|_F$ denotes the Frobenius norm. We notice that

$$\|\tilde{\mathbf{W}} - \mathbf{G} + \mathbf{G} - \mathbf{F} \mathbf{F}^\mathsf{T}\|_F \leq \|\tilde{\mathbf{W}} - \mathbf{G}\|_F + \|\mathbf{G} - \mathbf{F} \mathbf{F}^\mathsf{T}\|_F, \tag{4}$$

for any $\mathbf{G} \in \mathbb{R}^{n \times n}$. Our goal is to minimize the LHS (left-hand side); Instead, we can minimize the RHS which is the upper-bound of LHS.

Thus we need to find the intermediate matrix $\mathbf{G}$, i.e., we learn a *surrogate graph* which is close but not identical to $\tilde{\mathbf{W}}$. Our *upper-bounding approach* offers flexibility which allows us to impose certain desirable properties. Note that matrix $\mathbf{F} \mathbf{F}^\mathsf{T}$ holds the following properties: (P1) symmetric, (P2) nonnegative, (P3) low rank, and (P4) positive semidefinite. This suggests a *convex* graph learning

$$\min_{\mathbf{G}} \quad \|\mathbf{G} - \tilde{\mathbf{W}}\|_F^2 \quad s.t. \quad \mathbf{G} \succcurlyeq 0, \quad \|\mathbf{G}\|_* \leq c, \mathbf{G} = \mathbf{G}^\mathsf{T}, \quad \mathbf{G} \geq 0, \tag{5}$$

where $\mathbf{G} \succcurlyeq 0$ denotes the positive semidefinite constraint, $\| \cdot \|_*$ denotes the trace norm, *i.e.* the sum of the singular values [8], and $c$ is a model parameter which controls the rank of $\mathbf{G}$. The constraint of $\mathbf{G} \geq 0$ is to force the similarity to be naturally non-negative. By intuition, one might impose row rank constraint of $\text{rank}(\mathbf{G}) \leq c$. But this leads to a *non-convex* optimization, which is undesirable in unsupervised and semi-supervised learning. Following matrix completion methods [5], the trace constraint in Eq. (5) is a good surrogate for the low rank constraint. For notational convenience, the normalized similarity matrix $\tilde{\mathbf{W}}$ is denoted as $\mathbf{W}$ in the rest of the paper.

By solving Eq. (5), we are actually seeking a similarity matrix which satisfies all the properties of a perfect similarity matrix (P1–P4) and which is close to the original input matrix $\mathbf{G}$. Our whole paper is here dedicated to solve Eq. (5) and to demonstrate the usefulness of its optimal solution in clustering and semi-supervised learning using both theoretical and empirical evidences.

## 1.2 Related Work

Our method can be viewed as a preprocessing for similarity matrix and a large number of machine learning and data mining approaches require a similarity matrix (interpreted as a weighted graph) as input. For example, in unsupervised clustering, Normalized Cut [19], Ratio Cut [11], Cheeger Cut [3] have been widely applied in various real world applications. In graphical models for relational data, *e.g.* Mixed Membership Block models [1] can be also interpreted as generative models on the similarity matrices among objects. Thus a similarity matrix preprocessing model can be widely applied.

A large number of approaches have been developed to learn similarity matrix with different emphasis. Local Linear Embedding (LLE ) [17, 18] and Linear Label Propagation [21] can be viewed as obtaining a similarity matrix using sparse coding. Another way to perform the similarity matrix preprocessing is to take a graph as input and to obtain a refined graph by learning, such as bi-stochastic graph learning [13]. Our method falls in this category. We will compare our method with these methods in the experimental section.

On the optimization techniques for problems with multiple constraints, there also exist many related researches. First, von Neumann provided a convergence proof of successive projection method that it guarantees to converge to feasible solution in convex optimization with multiple constraints, which was employed in the paper by Liu *et al.* [13]. In this paper, we develop a novel optimization algorithm to solve the optimization problem with multiple convex constraints (including the inequality constraints), which is guaranteed to find the global solution. More explicitly, we develop a variant of inexact Augmented Lagrangian Multiplier method to handle inequality constraints. We also develop a useful Lemma to handle the subproblems with trace norm constraint in the main algorithm. Interestingly, one of the derived subproblems is the $\ell_1$ ball projection problem, which can be solved elegantly by simple thresholding.

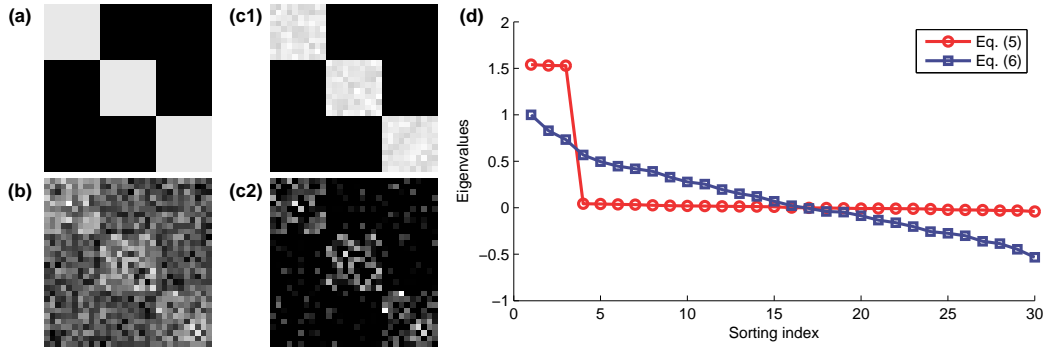

Figure 1: A toy example of low rank and positive semidefinite graph learning. **(a)**: A perfect similarity matrix. **(b)**: Adding noise from (a). (c1): the optimal solution of Eq. (5) by using the matrix in (b) as **G**. (c2): the optimal solution of Eq. (6) by using the matrix in (b) as **G**. (d): sorted eigenvalues for the two solutions of Eq. (5) and Eq. (6).

## 2 A Toy Example

We first emphasize the usefulness of the positive semidefinite and low rank constraints in problem of Eq. (5) using a toy example. In this toy example, we also solve the following problem for contrast,

$$\min_{\mathbf{G}} \; \|\mathbf{G} - \mathbf{W}\|_F^2 \;\; s.t. \; \mathbf{G} = \mathbf{G}^\mathsf{T}, \;\; \mathbf{Ge} = \mathbf{e}, \;\; \mathbf{G} \geq 0, \tag{6}$$

where $\mathbf{e} = [1, 1, \cdots, 1]^T$ and the constraints of positive semidefinite and low rank are removed from Eq. (5) and instead a bi-stochastic constraint is applied ($\mathbf{Ge} = \mathbf{e}$). Notice that the model defined in Eq. (6) is used in the bi-stochastic graph learning [13]. We solve Eqs. (5) and (6) for the same input **G** and compare the solution to see the effect of positive semidefinite and low rank constraints.

In the toy example, we first generate a perfect similarity matrix **W** in which $W_{ij} = 1$ if data points $i, j$ are in the same group and $W_{ij} = 0$ otherwise. Three groups of data points (10 data points in each group) are considered. **G** are shown in Figure 1 **(a)** with black color denoting zeros values. We then randomly add some positive noise on **G** which is shown in Figure 1 **(b)**. Then we solve Eqs. (5) and (6) and the results of **G** is shown in Figure 1 **(c1)** and **(c2)**. The observation is that Eq. (5) recover the perfect similarity much more accurately than Eq. (6). The reason is that in model of Eq. (6), the low rank and positive semidefinite constraints are ignored and the result deviates from the ground truth.

We show the eigenvalues distributions of **G** for solution in Figure 1 **(d)** for both methods in Eqs. (5) and (6). One can observe that the solution Eq. (5) gives low rank and positive semidefinite results, while the solution for Eq. (6) is full rank and has negative eigenvalues.

Since the solution of Eq. (5) is always positive, symmetric, low rank, and positive semidefinite, we called our solution the Non-negative Low-rank Kernel (NLK).

### 2.1 NLK for Semi-supervised Learning

Although NKL is mainly developed for unsupervised learning, it can be easily extended to incorporate the label information in semi-supervised learning [23]. Assume we are given a set of data $\mathbf{X} = [\mathbf{x}_1, \mathbf{x}_2, \cdots, \mathbf{x}_\ell, \mathbf{x}_{\ell+1}, \cdots, \mathbf{x}_n]$ where the first $l$ data points are labeled as $[y_1, y_2, \cdots, y_\ell]$. Then we have more information to learn a better similarity matrix. Here we add additional constraints on Eq. (5) by enforcing the similarity to be zeros for those paris of data points in different classes, *i.e.* $\mathbf{G}_{ij} = 0$ if $y_i \neq y_j, 1 \leq i, j \leq \ell$. By considering all the constraints, we optimize the following,

$$\min_{\mathbf{G}} \; \|\mathbf{G} - \mathbf{W}\|_F^2 \;\; s.t. \; \mathbf{G} \succcurlyeq 0, \;\; \|\mathbf{G}\|_* \leq c, \mathbf{G} = \mathbf{G}^\mathsf{T}, \;\; \mathbf{G} \geq 0, \; \mathbf{G}_{ij} = 0, \forall y_i \neq y_j. \tag{7}$$

We will demonstrate the advantage of these semi-supervision constraints in the experimental section. The computational algorithm is given in §3.3.

# 3 Optimization

The optimization problemss in Eqs. (5) and (7) are non-trivial since there are multiple constraints, including both equality and inequality constraints. Our strategy is to introduce two extra copies of optimization variable $\mathbf{X}$ and $\mathbf{Y}$ to split the constraints into several directly solvable subproblems,

$$\min_{\mathbf{G}} \quad \|\mathbf{G} - \mathbf{W}\|_F^2, \quad s.t. \quad \mathbf{G} \geq \mathbf{0} \tag{8a}$$

$$\min_{\mathbf{X}} \quad \|\mathbf{X} - \mathbf{W}\|_F^2, \quad s.t. \quad \mathbf{X} \succcurlyeq 0, \quad \text{with} \quad \mathbf{X} = \mathbf{G}. \tag{8b}$$

$$\min_{\mathbf{Y}} \quad \|\mathbf{Y} - \mathbf{W}\|_F^2, \quad s.t. \quad \|\mathbf{Y}\|_* \leq c, \quad \text{with} \quad \mathbf{Y} = \mathbf{G} \tag{8c}$$

More formally, we solve the following problem,

$$\min_{\mathbf{G}} \quad \|\mathbf{G} - \mathbf{W}\|_F^2 \tag{9a}$$

$$s.t. \quad \mathbf{G} \geq \mathbf{0} \tag{9b}$$

$$\mathbf{X} = \mathbf{G}, \quad \mathbf{X} \succcurlyeq 0, \tag{9c}$$

$$\mathbf{Y} = \mathbf{G}, \quad \|\mathbf{Y}\|_* \leq c, \tag{9d}$$

One should notice that problem in Eqs. (9a) – (9d) is equivalent to our main problem in Eq. (5). In the rest of this section, we will employ a variant of Augmented Lagrangian Multiplier (AML) method to solve Eqs. (9a) – (9c).

## 3.1 Seeking Global Solutions: A Variant of ALM

The augmented Lagrangian multiplier function of Eqs. (9a) – (9d) is

$$\Phi(\mathbf{G}, \mathbf{X}, \mathbf{Y}) = \|\mathbf{G} - \mathbf{W}\|_F^2 - \langle \Lambda, \mathbf{X} - \mathbf{G} \rangle + \frac{\mu}{2}\|\mathbf{G} - \mathbf{X}\|_F^2 - \langle \Sigma, \mathbf{Y} - \mathbf{G} \rangle + \frac{\mu}{2}\|\mathbf{G} - \mathbf{Y}\|_F^2, \tag{10}$$

with constraints of $\mathbf{G} \geq \mathbf{0}, \mathbf{X} \succcurlyeq 0$, and $\|\mathbf{Y}\|_* \leq c$, where $\Lambda, \Sigma$ are the Lagrangian multipliers.

Then ALM method leads to the following updating steps,

$$\mathbf{G} \quad \leftarrow \quad \arg\min_{\mathbf{G} \geq 0} \Phi(\mathbf{G}, \mathbf{X}, \mathbf{Y}, \mathbf{Z}) \tag{11a}$$

$$\mathbf{X} \quad \leftarrow \quad \arg\min_{\mathbf{X} \succcurlyeq 0} \Phi(\mathbf{G}, \mathbf{X}, \mathbf{Y}, \mathbf{Z}) \tag{11b}$$

$$\mathbf{Y} \quad \leftarrow \quad \arg\min_{\|\mathbf{Y}\|_* \leq c} \Phi(\mathbf{G}, \mathbf{X}, \mathbf{Y}, \mathbf{Z}) \tag{11c}$$

$$\Lambda \quad \leftarrow \quad \Lambda - \mu\,(\mathbf{G} - \mathbf{X}) \tag{11d}$$

$$\Sigma \quad \leftarrow \quad \Sigma - \mu\,(\mathbf{G} - \mathbf{Y}) \tag{11e}$$

$$\mu \quad \leftarrow \quad \gamma\mu, \quad t \leftarrow t + 1. \tag{11f}$$

Notice that the symmetric constraint is removed here. We will later show that given a symmetric input $\mathbf{W}$, the output of our algorithm automatically satisfies the symmetric constraints.

## 3.2 Solving the Subproblems in ALM

$\mathbf{X}$ and $\mathbf{Y}$ updating algorithms in Eqs. (11b and 11c) contain eigenvalue constraints, which appear complicated. Fortunately they have closed form solution. To show that, we first introduce the following useful Lemma.

**Lemma 3.1.** *Consider the following problem,*

$$\min_{\mathbf{X}} \|\mathbf{X} - \mathbf{A}\|_F^2, \ s.t. \ \phi_i(\mathbf{X}) \leq c_i, 1 \leq i \leq m, \tag{12}$$

*where $\phi_i(\mathbf{X}) \leq c_i$ is any constraint on eigenvalues of $\mathbf{X}$, $i = 1, 2, \cdots, m$ and $m$ is the number of constraints. Then there exists a diagonal matrix $\mathbf{S}$ such that $\mathbf{U}\mathbf{S}\mathbf{U}^\mathsf{T}$ is an optimizer of Eq. (12), where $\mathbf{U}\Sigma\mathbf{U}^\mathsf{T} = \mathbf{A}$ is the eigenvector decomposition of $\mathbf{A}$. $\mathbf{S}$ relates to eigenvalues of $\Sigma = diag(\lambda_1, \cdots, \lambda_n)$ and satisfying the constraints.*

*Proof.* Let $\mathbf{VSV^\intercal} = \mathbf{X}$ and $\mathbf{UDU^\intercal} = \mathbf{A}$ be the eigenvector decomposition of $\mathbf{X}$ and $\mathbf{A}$, respectively. By applying von Neumann's trace inequality, the following holds for any $\mathbf{X}$ and $\mathbf{A}$,

$$\mathbf{tr X^\intercal A} \leq \mathbf{tr SD}. \tag{13}$$

Then

$$\mathbf{tr VSV^\intercal A} = \mathbf{tr X^\intercal A} \leq \mathbf{tr SD} = \mathbf{tr(USU^\intercal)(UDU^\intercal)} = \mathbf{USU^\intercal A}, \tag{14}$$

which leads to

$$\|\mathbf{USU^\intercal - A}\|_F^2 \leq \|\mathbf{VSV^\intercal - A}\|_F^2. \tag{15}$$

Now assume $\mathbf{X} = \mathbf{VSV^\intercal}$ is a minimizer of Eq. (12). By comparing two solutions of $\mathbf{X} = \mathbf{VSV^\intercal}$ and $\mathbf{Z} = \mathbf{USU^\intercal}$, one should notice (a) that $\mathbf{Z}$ satisfies all the constraints of $\phi_i(\mathbf{Z}) = \phi_i(\mathbf{X}) \leq c_i, 1 \leq i \leq m$ in Eq. (12) and (b) that $\mathbf{Z}$ gives equal or less value of the objective, thus $\mathbf{Z} = \mathbf{USU^\intercal}$ ia also a minimizer of Eq. (12). $\qquad\square$

Lemma 3.1 shows an interesting property of the matrix approximation with eigenvalue or singular value constraint: the optimal solution matrix shares the same subspace of the input matrix. This is useful, because once the subspace is determined, the whole optimization becomes much easier. Thus the lemma provides a powerful mathematical tool in computation of optimization problem with eigenvalue and singular value constraints. Here, we apply Lemma 3.1 to solve the updating of $\mathbf{X}$ and $\mathbf{Y}$ in §3.2.2 - 3.2.3.

### 3.2.1 Updating G

By ignoring the irrelevant terms with respect to $\mathbf{G}$ , we can rewrite Eq. (11a) as following,

$$\mathbf{G} \quad \leftarrow \quad \arg\min_{\mathbf{G} \geq 0} \|(2 + 2\mu)\mathbf{G} - (2\mathbf{W} + \mu(\mathbf{X} + \mathbf{Y}) + \Lambda + \Sigma)\|_F^2 + \text{const} \tag{16}$$

$$= \quad \max\left(\frac{2\mathbf{W} + \mu(\mathbf{X} + \mathbf{Y}) + \Lambda + \Sigma}{2 + 2\mu}, 0\right). \tag{17}$$

### 3.2.2 Updating X

For Eq. (11b), we need to solve the following subproblem

$$\min_{\mathbf{X}} \|\mathbf{X} - \mathbf{P}\|_F^2, \quad \mathbf{X} \succcurlyeq 0, \quad \text{where} \quad \mathbf{P} = \mathbf{G} + \Lambda/\mu. \tag{18}$$

Notice that $\mathbf{X} \succcurlyeq 0$ is constraint on the eigenvalues of $\mathbf{X}$. Then we can directly apply Lemma 3.1, $\mathbf{X}$ can be written as $\mathbf{USU^\intercal}$ and Eq. (18) becomes

$$\min_{\mathbf{S}} \|\mathbf{USU^\intercal - UDU^\intercal}\|_F^2, \quad s.t. \ \mathbf{S} \geq 0, \tag{19}$$

where $\mathbf{UDU^\intercal} = \mathbf{P}$ is the eigenvector decomposition of $\mathbf{P}$. Let $\mathbf{S} = \mathbf{diag}(s_1, s_2, \cdots, s_n)$ and $\mathbf{D} = \mathbf{diag}(d_1, d_2, \cdots, d_n)$. Then Eq. (19) can be further rewritten as,

$$\min_{s_1, s_2, \cdots, s_n} \sum_{i=1}^{n} (s_i - d_i)^2, \quad s.t. \ s_i \geq 0, i = 1, 2, \cdots, n. \tag{20}$$

Eq. (20) has simple closed form solution as $s_i = \max(d_i, 0), i = 1, 2, \cdots, n$.

### 3.2.3 Updating Y

Eq. (11c) can be rewritten as,

$$\min_{\mathbf{Y}} \|\mathbf{Y} - \mathbf{Q}\|_F^2, \quad \|\mathbf{Y}\|_* \leq c, \tag{21}$$

where $\mathbf{Q} = \mathbf{G} + \frac{1}{\mu}\Sigma$. The corresponding Lagrangian function is,

$$\mathcal{L}(\mathbf{Y}, \lambda) = \|\mathbf{Y} - \mathbf{Q}\|_F^2 + \lambda \left(\|\mathbf{Y}\|_* - c\right). \tag{22}$$

Since we do not know the true Lagrangian multiplier $\lambda$, we cannot directly apply the singular value thresholding technique [4]. However, we find Lemma 3.1 useful again. We notice that $\mathbf{Y}$ is symmetric and the constraint of $\|\mathbf{Y}\|_* \leq c$ becomes a constraint on the eigenvalues of $\mathbf{Y}$. Let $\mathbf{Y} = \mathbf{USU^\intercal}$ and by directly applying Lemma 3.1, Eq. (21) can be further written as,

$$\min_{\mathbf{S}} \|\mathbf{USU^\intercal - UDU^\intercal}\|_F^2, \quad s.t. \ \sum_{i=1}^{n} |s_i| \leq c, \tag{23}$$

or,

$$\min_{\mathbf{s}} \|\mathbf{s} - \mathbf{d}\|^2, \quad s.t. \quad \sum_{i=1}^{n} |s_i| \leq c, \tag{24}$$

where $\mathbf{S} = \mathbf{diag}(\mathbf{s}), \mathbf{s} = [s_1, s_2, \cdots, s_n]^\mathsf{T}, \mathbf{D} = \mathbf{diag}(\mathbf{d})$, and $\mathbf{d} = [d_1, d2, \cdots, d_n]^\mathsf{T}$.

Interestingly, the above problem is a standard $\ell_1$ ball optimization problem which has been studied for a long time and Duchi *et al.* has recently provided a simple and elegant solution [7]. The final solution is to search the least $\theta \geq 0$ such that $\sum_i \max(|d_i| - \theta, 0) \leq c$, *i.e.*

$$\theta = \arg\min_{\theta} \theta \quad s.t. \quad \sum_{i=1}^{n} \max(|d_i| - \theta, 0) \leq c. \tag{25}$$

This can be easily done by sorting the $|d_i|$ and try the $\theta$ values between two consecutive sorted $|d_i|$. And the solution is

$$s_i = \mathbf{sign}(d_i) \max(|d_i| - \theta, 0). \tag{26}$$

Notice that in each step of algorithm, the solution has closed form solution and that the output of $\mathbf{G}$ is always symmetric, which indicates that the constraint of $\mathbf{G} = \mathbf{G}^\mathsf{T}$ is automatically satisfied in each step.

### 3.3 NLK Algorithm For Semi-supervised Learning

In many real world settings, we know partially of data class labels and hope to further utilize such information, as described in Eq. (7). Fortunately, the corresponding optimization problem remains convex. The augmented Lagrangian multiplier function is

$$\begin{aligned} \Phi(\mathbf{G}, \mathbf{X}, \mathbf{Y}) = \quad & \|\mathbf{G} - \mathbf{W}\|_F^2 - \langle \Lambda, \mathbf{X} - \mathbf{G} \rangle + \tfrac{\mu}{2}\|\mathbf{X} - \mathbf{G}\|_F^2 - \langle \Sigma, \mathbf{Y} - \mathbf{G} \rangle \\ & + \tfrac{\mu}{2}\|\mathbf{Y} - \mathbf{G}\|_F^2 + \sum_{(i,j)\in T} \left( \tfrac{\mu}{2} G_{ij}^2 - \Omega_{ij} G_{ij} \right), \end{aligned} \tag{27}$$

This is identical to Eq. (10), except we added $\Omega$ as additional Lagrangian multiplier for the semi-supervised constraints, i.e. the desired similarity $G_{ij} = 0$ for $(i, j)$ having different known class labels. Here $T = \{(i, j) : y_i \neq y_j, i, j = 1, 2, \cdots, \ell\}$.

We modify Algorithm of Eqs. (11a–11f) to solve this problem. The updating of $\mathbf{X}$ and $\mathbf{Y}$ remains the same as NLK algorithm described previously. To update $\mathbf{G}$, we set $\partial\Phi(\mathbf{G}, \mathbf{X}, \mathbf{Y})/\partial\mathbf{G} = 0$ and obtain

$$G_{ij} \leftarrow \begin{cases} \max\left( \dfrac{2W_{ij} + \mu(X_{ij} + Y_{ij}) + \Lambda_{ij} + \Sigma_{ij} + \Omega_{ij}}{2 + 3\mu}, 0 \right) & \text{if } y_i \neq y_j, \\[4mm] \max\left( \dfrac{2W_{ij} + \mu(X_{ij} + Y_{ij}) + \Lambda_{ij} + \Sigma_{ij}}{2 + 2\mu}, 0 \right) & \text{otherwise.} \end{cases} \tag{28}$$

For Lagrangian multiplier $\Omega$, the corresponding updating is

$$\Omega_{ij} \leftarrow \Omega_{ij} - \mu G_{ij}, \forall y_i \neq y_j. \tag{29}$$

Thus the semi-supervised learning algorithm is nearly identical to the unsupervised learning algorithm — one strength of our unified NLK approach.

We summarize the NLK algorithms for unsupervised and semi-supervised learning in Algorithm 1. In the algorithm, Lines 4 and 9 are updated for semi-supervised learning while other lines are shared.

**Algorithm 1** NLK Algorithm For Supervised Learning and Semi-supervised Learning

---

**Require:** Weighted graph $\mathbf{W}$, model parameters $c$, optimization parameter $\gamma$, partial label $\mathbf{y}$ for semi-supervised learning.

1: **Initialization**: $\mathbf{G} = \mathbf{W}, \Lambda = 0, \Sigma = 0, \Omega = 0, \mu = 1$.
2: **while** Not converged **do**
3:      For unsupervised learning, $\mathbf{G} \leftarrow \max\left(\frac{2\mathbf{W}+\mu(\mathbf{X}+\mathbf{Y})+\Lambda+\Sigma}{2+2\mu}, 0\right)$.
4:      For semi-supervised learning, update $\mathbf{G}$ using Eq. (28).
5:      $\mathbf{X} \leftarrow \mathbf{UD_+U^\intercal}$ where $\mathbf{UDU^\intercal} = \mathbf{G} + \Lambda/\mu$.
6:      $\mathbf{Y} \leftarrow \mathbf{USU^\intercal}$ where $\mathbf{UDU^\intercal} = \mathbf{G} + \Sigma/\mu$ and $\mathbf{S}$ is computed by Eq. (26).
7:      $\Lambda \leftarrow \Lambda - \mu(\mathbf{X} - \mathbf{G})$
8:      $\Sigma \leftarrow \Sigma - \mu(\mathbf{Y} - \mathbf{G})$
9:      For semi-supervised learning, $\Omega_{ij} \leftarrow \Omega_{ij} - \mu G_{ij}, \forall y_i \neq y_j$.
10:     $\mu \leftarrow \gamma\mu$.
11: **end while**
12: **return** $\mathbf{G}$

---

### 3.4 Theoretical Analysis of The Algorithm

Since the objective function and all the constraints are convex, we have the following [12]

**Theorem 3.2.** *Algorithm 1 converges to the global solution of Eq. (5) or Eq. (7).*

Notice that this conclusion is stronger than that in the related research papers [13] for graph learning.

## 4 Experimental Validation

As mentioned in the introduction section, the optimization results for NLK (Eq. (5)) can be used as preprocessing for any graph based methods. Here we evaluated NLK on several state-of-the-art graph based learning models, include Normalized Cut (Ncut) [19] for unsupervised learning and Gaussian Fields and Harmonic Functions (GFHF) and local and global consistency learning (LGC) for semi-supervised learning. We compare the clustering in both clustering accuracy and normalized mutual information (NMI). For the semi-supervised learning model (Eq. (7)), we evaluate the our models on GFHF and LGC models. For semi-supervised learning, we measure the classification accuracy. We verify the algorithms on four data sets: AT&T ($n = 400, p = 644, K = 40$), BinAlpha ($n = 1404, p = 320, K = 36$), Segment ($n = 2310, p = 19, K = 7$), and Vehicle($n = 946, p = 18, K = 4$) from UCI data [9], where $n, p,$ and $K$ are the number of data points, features, and classes, respectively.

### 4.1 Experimental Settings

For clustering, we compare three similarity matrices: (1) original from Gaussian kernel matrix, $w_{ij} = \exp\left(-\|\mathbf{x}_i - \mathbf{x}_j\|^2/2\sigma^2\right)$, where $\sigma$ is set to the average pairwise distances among all the data points. (2) the BBS (Bregmanian Bi-Stochastication) [20], and our method (NLK). The clustering algorithm of Normalized Cut [19] is applied on the three similarity matrices. Then we have total three clustering approaches: Normalized Cut (Ncut), BBS+Ncut, and NLK+Ncut. For each clustering method, we try 100 random trials for different clustering initializations. For the semi-supervised learning, we test three basic graph-based semi-supervised learning models. Gaussian Fields and Harmonic Functions (GFHF) [23], Local and Global Consistency learning (LGC) [22], and Green's function (Green) [6]. We compare 4 types of similarity matrices: original Gaussian kernel matrix, as discussed before, BBS, NLK, and NLK with semi-supervised constraints (model in Eq. (7), denoted by NLK_Semi). Then we totally have $3 \times 4$ methods. For each method, we random split the data to 30%/70% where 30% is is used as labeled data an the other 70% as the testing data. We try 100 random split and we report the average and standard deviations.

### 4.2 Parameter Settings

For all the similarity learning approaches (BBS, NLK, and NLK_Semi), we set the convergent criteria as follows. If $\|\mathbf{G}^{t+1} - \mathbf{G}^t\|_F^2/\|\mathbf{G}^t\|_F^2 < 10^{-10}$ we stop the algorithms. For our methods (NLK

Table 1: Clustering accuracy and NMI comparison over 3 methods, Normalized Cut (Ncut), BBS+Ncut, and NLK+Ncut on 4 data sets. The best results are highlighted in bold.

| | Accuracy | | | NMI | | |
|---|---|---|---|---|---|---|
| | Ncut | BBS+Ncut | NLK+Ncut | Ncut | BBS+Ncut | NLK+Ncut |
| AT&T | $0.607 \pm 0.022$ | $0.686 \pm 0.021$ | $\mathbf{0.767 \pm 0.006}$ | $0.785 \pm 0.025$ | $0.836 \pm 0.026$ | $\mathbf{0.873 \pm 0.025}$ |
| BinAlpha | $0.431 \pm 0.018$ | $0.444 \pm 0.022$ | $\mathbf{0.490 \pm 0.009}$ | $0.618 \pm 0.013$ | $0.629 \pm 0.015$ | $\mathbf{0.673 \pm 0.011}$ |
| Segment | $0.613 \pm 0.018$ | $0.593 \pm 0.009$ | $\mathbf{0.616 \pm 0.002}$ | $0.528 \pm 0.016$ | $\mathbf{0.579 \pm 0.013}$ | $0.538 \pm 0.002$ |
| Vehicle | $0.383 \pm 0.001$ | $0.383 \pm 0.000$ | $\mathbf{0.426 \pm 0.000}$ | $0.121 \pm 0.001$ | $0.122 \pm 0.000$ | $\mathbf{0.184 \pm 0.000}$ |

and NLK_Semi), there is one model parameter $c$, which is always set to be $c = 0.5\|\mathbf{W}\|_*$ where $\mathbf{W}$ is the input similarity matrix.

### 4.3 Experimental Results

We show that clustering results in Table 1 where we compare both measurements (accuracy, NMI) over 3 methods on 4 data sets. For each method, we report the average performance and the corresponding standard deviation. Out of 4 data sets, our method outperforms all the other methods with all the measurements on 3 data sets (AT&T, BinAlpha, and Vehicle).

We also test the semi-supervised learning performance over the 12 methods on 4 data sets. In each method on each data, we show the original performance values with dots. Shown are also the average accuracies and the corresponding standard deviations. Out of 4 data sets, our method (NLK and NLK_Semi) outperform the other methods.

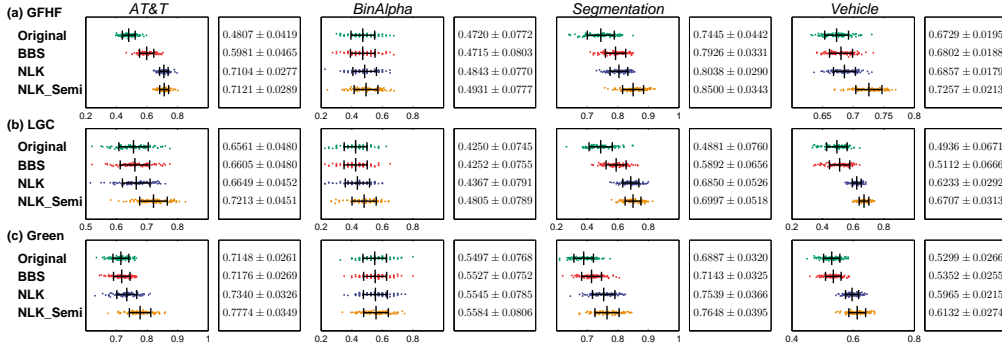

Figure 2: Semi-supervised learning performance over the 12 methods on 4 data sets. Original accuracy value for each random split is plotted with dots. Shown are also the average accuracies and the corresponding standard deviations.

## 5 Conclusions and Discussion

In this paper, we derive a similarity learning model based on convex optimizations. We demonstrate that the low rank and positive semidefinite constraints are nature in the similarity. Further more, we develop new sufficient algorithm to obtain global solution with theoretical guarantees. We also develop more optimization techniques that are potentially useful in the related eigenvalues or singular values constraints optimization. The presented model is verified on extensive experiments, and the results show that our method enhances the quality of the similarity matrix significantly, in both clustering and semi-supervised learning.

**Acknowledgement** This research was partially supported by NSF-CCF 0830780, NSF-DMS 0915228, NSF-CCF 0917274, NSF-IIS 1117965.

# References

[1] E. Airoldi, D. Blei, E. Xing, and S. Fienberg. A latent mixed membership model for relational data. In *Proceedings of the 3rd international workshop on Link discovery*, pages 82–89. ACM, 2005.

[2] M. Belkin and P. Niyogi. Laplacian eigenmaps for dimensionality reduction and data representation. *Neural computation*, 15(6):1373–1396, 2003.

[3] T. Bühler and M. Hein. Spectral Clustering based on the graph p-Laplacian. In *Proceedings of the 26th Annual International Conference on Machine Learning*, pages 81–88. ACM, 2009.

[4] J. Cai, E. Candes, and Z. Shen. A singular value thresholding algorithm for matrix completion. *IEEE Trans. Inform. Theory, 56(5), 2053-2080*, (5):2053–2080, 2008.

[5] E. Candes and Y. Plan. Matrix completion with noise. *Proceedings of the IEEE*, 98(6):925–936, 2010.

[6] C. Ding, R. Jin, T. Li, and H. Simon. A learning framework using Green's function and kernel regularization with application to recommender system. In *Proceedings of the 13th ACM SIGKDD international conference on Knowledge discovery and data mining*, pages 260–269. ACM, 2007.

[7] J. Duchi, S. Shalev-Shwartz, Y. Singer, and T. Chandra. Efficient projections onto the l 1-ball for learning in high dimensions. In *Proceedings of the 25th international conference on Machine learning*, pages 272–279. ACM, 2008.

[8] M. Fazel. *Matrix rank minimization with applications*. PhD thesis, Stanford University, 2002.

[9] A. Frank and A. Asuncion. UCI machine learning repository, 2010.

[10] M. Gu, H. Zha, C. Ding, X. He, H. Simon, and J. Xia. Spectral relaxation models and structure analysis for k-way graph clustering and bi-clustering. *UC Berkeley Math Dept Tech Report*, 2001.

[11] L. Hagen and A. Kahng. New spectral methods for ratio cut partitioning and clustering. *Computer-Aided Design of Integrated Circuits and Systems, IEEE Transactions on*, 11(9):1074–1085, 2002.

[12] R. Lewis, V. Torczon, and L. R. Center. A globally convergent augmented lagrangian pattern search algorithm for optimization with general constraints and simple bounds. *SIAM Journal on Optimization*, 12(4):1075–1089, 2002.

[13] W. Liu and S. Chang. Robust multi-class transductive learning with graphs. 2009.

[14] D. Luo, C. Ding, and H. Huang. Graph evolution via social diffusion processes. *Machine Learning and Knowledge Discovery in Databases*, pages 390–404, 2011.

[15] D. Luo, C. Ding, F. Nie, and H. Huang. Cauchy graph embedding. *ICML2011*, pages 553–560, 2011.

[16] A. Ng, M. Jordan, and Y. Weiss. On spectral clustering: Analysis and an algorithm. *Advances in neural information processing systems*, 2:849–856, 2002.

[17] S. Roweis and L. Saul. Nonlinear dimensionality reduction by locally linear embedding. *Science*, 290(5500):2323, 2000.

[18] H. Seung and D. Lee. The manifold ways of perception. *Science(Washington)*, 290(5500):2268–9, 2000.

[19] J. Shi and J. Malik. Normalized cuts and image segmentation. *Pattern Analysis and Machine Intelligence, IEEE Transactions on*, 22(8):888–905, 2002.

[20] F. Wang, P. Li, and A. König. Learning a Bi-Stochastic Data Similarity Matrix. In *2010 IEEE International Conference on Data Mining*, pages 551–560. IEEE, 2010.

[21] F. Wang and C. Zhang. Label propagation through linear neighborhoods. *IEEE Transactions on Knowledge and Data Engineering*, pages 55–67, 2007.

[22] D. Zhou, O. Bousquet, T. Lal, J. Weston, and B. Schölkopf. Learning with local and global consistency. In *Advances in Neural Information Processing Systems 16: Proceedings of the 2003 Conference*, pages 595–602, 2004.

[23] X. Zhu, Z. Ghahramani, and J. Lafferty. Semi-supervised learning using gaussian fields and harmonic functions. In *ICML 2003*.

